# Linear Submodular Bandits
## and their Application to Diversified Retrieval

**Yisong Yue**
iLab, Heinz College
Carnegie Mellon University
yisongyue@cmu.edu

**Carlos Guestrin**
Machine Learning Department
Carnegie Mellon University
guestrin@cs.cmu.edu

## Abstract

Diversified retrieval and online learning are two core research areas in the design of modern information retrieval systems. In this paper, we propose the *linear submodular bandits problem*, which is an online learning setting for optimizing a general class of feature-rich submodular utility models for diversified retrieval. We present an algorithm, called LSBGREEDY, and prove that it efficiently converges to a near-optimal model. As a case study, we applied our approach to the setting of personalized news recommendation, where the system must recommend small sets of news articles selected from tens of thousands of available articles each day. In a live user study, we found that LSBGREEDY significantly outperforms existing online learning approaches.

## 1   Introduction

User feedback has become an invaluable source of training data for optimizing information retrieval systems in a rapidly expanding range of domains, most notably content recommendation (e.g., news, movies, ads). When designing retrieval systems that adapt to user feedback, two important challenges arise. First, the system should recommend optimally diversified content that maximizes coverage of the information the user finds interesting (to maximize positive feedback). Second, the system should make exploratory recommendations in order to learn a reliable model from feedback.

*Challenge 1: diversification*. In most retrieval settings, the retrieval system must recommend sets of articles, rather than individual articles. Furthermore, the recommended articles should be well diversified. This is motivated by the principle that recommending redundant articles leads to diminishing returns on utility, since users need to consume redundant information only once. This notion of diminishing returns is well-captured by submodular utility models, which have become an increasingly popular approach to modeling diversified retrieval tasks in recent years [24, 25, 18, 3, 21, 9, 16].

*Challenge 2: feature-based exploration*. In most retrieval settings, users typically only provide feedback on the articles recommended to them. This partial feedback issue leads to an inherent tension between exploration and exploitation when deciding which articles to recommend to the user. Furthermore, it is typically desirable to learn a feature-based model that can generalize to new or previously unseen articles and users; this is often called the contextual bandits problem [13, 15, 7].

Although there exist approaches that have addressed these challenges individually, to our knowledge there is no single approach which solves both simultaneously and is also practical to implement. For instance, existing online approaches for optimizing submodular functions typically assume a feature-free model, and thus cannot generalize easily [18, 22, 23]. Such approaches measure performance relative to the single best set (e.g., of articles). Thus, they are not suitable for many retrieval settings since the set of available articles can change frequently (e.g., news recommendation).

In this paper, we address both challenges in a unified framework. We propose the *linear submodular bandits problem*, which is an online learning setting for optimizing a general class of feature-based

submodular utility models. To make learning practical, we represent the benefit of adding an article to an existing set of selected articles as a linear model with respect to the user's preferences. This class of models encompasses several existing information coverage utility models for diversified retrieval [24, 25, 9], and allows us to learn flexible models that can generalize to new predictions.

Similar to the contextual bandits setting considered in [15], our setting can be characterized as a feature-based exploration-exploitation problem, where the uncertainty lies in how best to model user interests using the available features. In contrast to [15], we aim to recommend optimally diversified sets of articles rather than just single articles. From that standpoint, modeling this additional layer of complexity in the bandit setting is our main technical contribution. We present an algorithm, called LSBGREEDY, to optimize this exploration-exploitation trade-off. When learning a $d$-dimensional model to recommend sets of $L$ articles for $T$ time steps, we prove that LSBGREEDY incurs regret that grows as $\mathcal{O}(d\sqrt{LT})$ (ignoring log factors). This regret matches the convergence rates of analogous algorithms for the conventional linear bandits setting [1, 20, 8].

As a case study, we applied our approach to the setting of personalized news recommendation [9, 15, 16]. In addition to simulation experiments, we conducted a live user study over a period of ten rounds, where in each round the retrieval system must recommend a small set of news articles selected from tens of thousands of available articles for that round. We compared against existing online learning approaches that either employ no exploration [9], or learn to recommend only single articles (and thus do not model diversity) [15]. Compared to previous approaches, we find that LSBGREEDY can significantly improve the performance of the retrieval system even when learning for a limited number of rounds. Our empirical results demonstrate the advantage of jointly tackling the challenges of diversification and feature-based exploration, as well as showcase the practicality of our approach.

## 2 Submodular Information Coverage Models

Before presenting our online learning setting, we first describe the class of utility functions that we optimize over. Throughout this paper, we use personalized news recommendation as our motivating example. In this setting, utility corresponds to the amount of interesting information covered by the set of recommended articles.

Suppose that news articles are represented using a set of $d$ "topics" or "concepts" that we wish to cover (e.g., the Middle East or the weather).[1] Intuitively, recommending two articles that cover highly overlapping topics might not be more beneficial than recommending just one of the articles – this is the notion of diminishing returns we wish to capture in our information coverage model.

Two key properties we will exploit are that our utility functions are monotone and submodular. A set function $F$ mapping sets of recommended articles $A$ to real values (e.g., the total information covered by $A$) is monotone and submodular if and only if

$$F(A \cup \{a\}) \geq F(A) \quad \text{and} \quad F(A \cup \{a\}) - F(A) \geq F(B \cup \{a\}) - F(B),$$

respectively, for all articles $a$ and sets $A \subseteq B$. In other words, since $A$ is smaller than $B$, the benefit of adding $a$ to $A$ is larger than the benefit of adding $a$ to $B$. Submodularity provides a natural framework for characterizing diminishing returns in information coverage, since the gain of adding a second (redundant) article on a topic will be smaller than the gain of adding the first.

For each topic $i$, let $F_i(A)$ be a monotone submodular function corresponding to how well the recommended articles $A$ cover topic $i$. We write the total utility of recommending $A$ as

$$F(A|w) = w^\top \langle F_1(A), \ldots, F_d(A) \rangle, \tag{1}$$

where $w \in \Re_+^d$ is a parameter vector indicating the user's interest level in each topic. Thus, $F(A|w)$ corresponds to the weighted information coverage of $A$, and depends on the preferences of the particular user. Since sums of monotone submodular functions are themselves monotone submodular, this implies that $F(A|w)$ is also monotone submodular (this would not hold if $w$ has negative components). When making recommendations, the goal then is to select the $A$ that maximizes $F(A|w)$. This class of information utility models encompasses several existing models of information coverage for diversified retrieval [24, 25, 9].

**Example: Probabilistic Coverage**. As an illustrative example, we now describe the probabilistic coverage model proposed in [9]. This will also be the coverage model used in our case study (see Section 5). Each article $a$ has some probability $P(i|a)$ of covering topic $i$.[2] Assuming each article $a \in A$ has an independent probability of covering each topic, then we can write $F_i(A)$ as

$$F_i(A) = 1 - \prod_{a \in A}(1 - P(i|a)), \tag{2}$$

which corresponds to the probability that topic $i$ is covered by at least one article in $A$. It is straightforward to check that $F_i$ in (2) is monotone submodular [9].

**Local Linearity**. One attractive property of $F(A|w)$ in (1) is that the incremental gains are *locally linear*. In particular, the incremental gain of adding $a$ to $A$ can be written as $w^\top \Delta(a|A)$, where

$$\Delta(a|A) = \langle\, F_1(A \cup \{a\}) - F_1(A)\, , \, \ldots\, , \, F_d(A \cup \{a\}) - F_d(A)\, \rangle. \tag{3}$$

In other words, the $i$-th component of $\Delta(a|A)$ corresponds to the incremental coverage (i.e., submodular advantage) of topic $i$ by article $a$, conditioned on articles $A$ having already been selected. This property will be exploited by our online learning algorithm presented in Section 4.

**Optimization**. Another attractive property of monotone submodular functions is that the myopic greedy algorithm is guaranteed to produce a near-optimal solution [17]. For any budget $L$ (e.g., $L = 10$ articles), the constrained optimization problem, $\mathrm{argmax}_{A:|A| \leq L} F(A|w)$, can be solved greedily to produce a solution that is within a factor $(1 - 1/e) \approx 0.63$ of optimal. Achieving better than $(1 - 1/e)OPT$ is known to be intractable unless $P = NP$ [10]. In practice, the greedy algorithm can often perform much better than this worst case guarantee (cf. [14]), and will be a central component in our online learning algorithm.

## 3 Problem Formulation

We propose the *linear submodular bandits problem* which is described in the following. At each time step $t = 1, \ldots, T$, our algorithm interacts with the user in the following way:

- A set of articles $\mathcal{A}_t$ is made available to the algorithm. Each article $a \in \mathcal{A}_t$ is represented using a set of $d$ basis coverage functions $F_1, \ldots, F_d$, defined as in Section 2, which is known to the algorithm.

- The algorithm chooses a ranked set of $L$ articles, denoted $A_t = (a_t^{(1)}, \ldots, a_t^{(L)})$, using the basis coverage functions of the articles and the outcomes of previous time steps.

- The user provides feedback (e.g., clicks on or ignores each article), and the rewards for each recommended articles $r_t(A_t)$ (4) are observed.

In order to develop our algorithm, we require a model of user behavior. We assume the user scans the recommended articles $A = (a^{(1)}, \ldots, a^{(L)})$ one by one in top-down fashion. For each article $a^{(\ell)}$, the user considers the *new information* covered by $a^{(\ell)}$ and not covered by the above articles $A^{(1:\ell-1)}$ ($A^{(1:\ell)}$ denotes the articles in the first $\ell$ slots). In our representation, this new information is $\Delta(a^{(\ell)}|A^{(1:\ell-1)})$ as in (3). The user then clicks on (or likes) $a^{(\ell)}$ with independent probability $(w^*)^\top \Delta(a^{(\ell)}|A^{(1:\ell-1)})$, where $w^*$ is the hidden preferences of the user. Formally, for any set of articles $A$ chosen at time $t$, the rewards $r_t(A)$ can be written as the sum of rewards at each slot,

$$r_t(A) = \sum_{\ell=1}^{L} r_t^{(\ell)}(A). \tag{4}$$

We assume each $r_t^{(\ell)}$ is an independent random variable bounded in $[0, 1]$ and satisfies

$$\mathbf{E}\left[r_t^{(\ell)}(A)\right] = (w^*)^\top \Delta(a^{(\ell)}|A^{(1:\ell-1)}), \tag{5}$$

where $w^*$ is a weight vector unknown to the algorithm with $\|w^*\| \leq S$. In other words, the expected reward in each slot is realizable, linear in $\Delta(a^{(\ell)}|A^{(1:\ell-1)})$, and independent of the other slots. We call this independence property *conditional submodular independence*, which we will leverage in

**Algorithm 1** LSBGREEDY

1: **input**: $\lambda$, $\alpha_t$
2: **for** $t = 1, \ldots, T$ **do**
3:     $M_t \leftarrow \lambda I_d + \sum_{\tau=1}^{t-1} \sum_{\ell=1}^{L} \Delta_\tau^{(\ell)} \left(\Delta_\tau^{(\ell)}\right)^\top$   *//covariance matrix*
4:     $b_t \leftarrow \sum_{\tau=1}^{t-1} \sum_{\ell=1}^{L} \hat{r}_\tau^{(\ell)} \Delta_\tau^{(\ell)}$   *//aggregate feedback so far*
5:     $w_t \leftarrow M_t^{-1} b_t$   *//linear regression using previous feedback as training data*
6:     $A_t \leftarrow \emptyset$
7:     **for** $\ell = 1, \ldots, L$ **do**
8:         $\forall a \in \mathcal{A}_t \setminus A(t) : \mu_a \leftarrow w_t^\top \Delta(a|A_t)$   *//compute mean estimate of utility gain*
9:         $\forall i \in \mathcal{A}_t \setminus A(t) : c_a \leftarrow \alpha_t \sqrt{\Delta(a|A_t)^\top M_t^{-1} \Delta(a|A_t)}$   *//compute confidence interval*
10:        set $a_t^{(\ell)} \leftarrow \arg\max_a (\mu_a + c_a)$   *//select article with highest upper confidence bound*
11:        store $\Delta_t^{(\ell)} \leftarrow \Delta\left(a_t^{(\ell)} \Big| A_t^{(1:\ell-1)}\right)$, $A_t \leftarrow A_t \cup \left\{a_t^{(\ell)}\right\}$
12:     **end for**
13:     recommend articles $A_t$ in the order selected, and observe rewards $\hat{r}_t^{(1)}, \ldots, \hat{r}_t^{(L)}$ for each slot
14: **end for**

---

our analysis. While conditional submodular independence may seem ideal, we will show in our user study experiments that it is not required for our proposed algorithm to achieve good performance.

Equations (4) and (5) imply that $\mathbf{E}[r_t(A)] = F(A|w^*)$ for $F$ defined as in (1). Thus, $\mathbf{E}[r_t]$ is monotone submodular, and a clairvoyant system with perfect knowledge of $w^*$ can greedily select articles to achieve (expected) reward at least $(1 - 1/e)OPT$, where $OPT$ denotes the total expected reward of the optimal recommendations for $t = 1, \ldots, T$. Let $A_t^*$ denote the optimal set of articles at time $t$. We quantify performance using the following notion of regret which we call *greedy regret*,

$$\text{Reg}_G(T) = \left(1 - \frac{1}{e}\right) \sum_{t=1}^{T} \mathbf{E}\left[r_t(A_t^*)\right] - \sum_{t=1}^{T} r_t(A_t) \equiv \left(1 - \frac{1}{e}\right) OPT - \sum_{t=1}^{T} r_t(A_t). \tag{6}$$

## 4 Algorithm and Main Results

A central question in the study of bandit problems is how best to balance the trade-off between exploration and exploitation (cf. [15]). To minimize regret (6), an algorithm must exploit its past experience to recommend sets of articles that appear to maximize information coverage. However, topics that appear good (i.e., interesting to the user) may actually be suboptimal due to imprecision in the algorithm's knowledge. In order to avoid this situation, the algorithm must explore by recommending articles about seemingly poor topics in order to gather more information about them.

In this section, we present an algorithm, called LSBGREEDY, which automatically trades off between exploration and exploitation (Algorithm 1). LSBGREEDY balances exploration and exploitation using upper confidence bounds on the estimated gain in utility, and builds upon upper confidence bound style algorithms for the conventional linear bandits setting [8, 20, 15, 7, 1]. Intuitively, the algorithm can be decomposed into the following components.

**Training a Model**. Since we employ a linear model, at each time $t$, we can fit an estimate $w_t$ of the true $w^*$ via linear regression on the previous feedback. Lines 3–5 in Algorithm 1 describe this step, where $\Delta_\tau^{(\ell)}$ denotes the incremental coverage features of the article selected at time $\tau$ and slot $\ell$, and $\hat{r}_\tau^{(\ell)}$ denotes the associated reward. Note that $\lambda$ in Line 3 is the standard regularization parameter.

**Estimating Incremental Coverage**. Given $w_t$, we can now estimate the incremental gain of adding any article $a$ to an existing set of results $A$. As discussed in Section 3, the true (expected) incremental gain is $(w^*)^\top \Delta(a|A)$. Our algorithm's estimate is $w_t^\top \Delta(a|A)$ (Line 8). If our algorithm were to purely exploit prior knowledge, then it would greedily choose articles that maximize $w_t^\top \Delta(a|A)$.[3]

**Computing Confidence Intervals**. Of course, each $w_t$ is an imprecise estimate of the true $w^*$. Given such uncertainty, a natural approach is to use confidence intervals which contain the true $w^*$

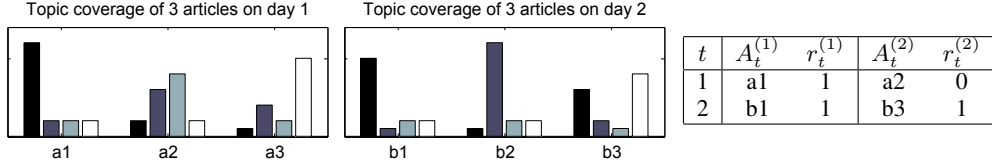

Figure 1: Illustrative example of LSBGREEDY for $L = 2$ and 2 days. Each day comprises 3 articles covering 4 topics, which are depicted in the two plots. Each row in the table describes the choices of LSBGREEDY and the resulting feedback. In day 1, LSBGREEDY recommends articles to explore topics 1, 2, and 3, and the user indicates liking a1 and disliking a2. In day 2, LSBGREEDY recommends b1 to exploitatively cover topic 1, and b3 to both cover topic 1 and explore topic 4.

with some target confidence (e.g., 95%). Our algorithm's uncertainty in the gain of article $a$ given set $A$ depends directly to how much feedback we have collected regarding prominent topics in $\Delta(a|A)$. In our linear setting, uncertainty is measured using the inverse covariance matrix $M_t^{-1}$ of the sub-modular features of the previously selected articles (Line 9). If our algorithm were to purely explore, then it would greedily select articles that have maximal uncertainty $\sqrt{\Delta(a|A)^\top M_t^{-1} \Delta(a|A)}$.

**Balancing Exploration and Exploitation**. In order to achieve low regret, LSBGREEDY greedily selects articles that maximize a compromise between estimated gain and uncertainty (Line 10), with $\alpha_t$ controlling the tradeoff. For any $\delta \in (0, 1)$, Lemma 3 in Appendix A.2 provides sufficient conditions on $\alpha_t$ for constructing confidence intervals,

$$w_t^\top \Delta(a|A) \pm \alpha_t \sqrt{\Delta(a|A)^\top M_t^{-1} \Delta(a|A)} \equiv w_t^\top \Delta(a|A) \pm \alpha_t \|\Delta(a|A)\|_{M_t^{-1}}, \qquad (7)$$

that contain the true value, $(w^*)^\top \Delta(a|A)$, with probability at least $1 - \delta$. In this sense, Line 10 maximizes the *upper confidence bound* on the true expected reward. Figure 1 provides an illustrative example of the behavior of LSBGREEDY.

We now state our main result, which essentially bounds the greedy regret (6) of LSBGREEDY as $\mathcal{O}(d\sqrt{TL})$ (ignoring log factors). This means that the average loss incurred per slot and per day by LSBGREEDY relative to $(1 - 1/e)OPT$ decreases at a rate of $\mathcal{O}(d/\sqrt{TL})$.

**Theorem 1.** *For $L \leq d$, $\lambda = L$, and $\alpha_t$ defined as*

$$\alpha_t = \sqrt{2 \log \left( 2 \det(M_t)^{1/2} \det(\lambda I_d)^{-1/2} / \delta \right)} + S\sqrt{\lambda}, \qquad (8)$$

*with probability at least $1 - \delta$, LSBGREEDY achieves greedy regret (6) bounded by*

$$Reg_G(T) \leq \alpha_T \sqrt{8TL \log \det(M_{T+1})} + \sqrt{2(1 + TL) \log \left( \frac{\sqrt{1 + TL}}{\delta/2} \right)} = \mathcal{O} \left( Sd\sqrt{TL} \log \left( \frac{TL}{\delta} \right) \right).$$

The proof of Theorem 1 is presented in Appendix A in the supplementary material. In practice, the choice of $\alpha_t$ in (8) may be overly conservative. As we show in our experiments, more aggressive choices of $\alpha_t$ can often lead to faster convergence.

## 5 Empirical Analysis: Case Study in News Recommendation

We applied LSBGREEDY to the setting of personalized news recommendation (cf. [9, 15, 16]), where the system is tasked with recommending sets of articles that maximally cover the interesting information of the available articles. The user provides feedback (e.g., by indicating that she likes or dislikes each article), and the goal is to maximize the total positive feedback by personalizing to the user. We conducted both simulation experiments as well as a live user study. Since real users are unlikely to behave exactly according to our modeling assumptions (e.g., obey conditional submodular independence), our user study tests the effectiveness of our approach in settings beyond those considered in our theoretical analysis.

### 5.1 Simulations

**Data**. We ran simulations using both synthetic datasets as well as the blog dataset from [9]. For each setting, we generated a hidden true preference vector $w^*$. For the synthetic data, all articles

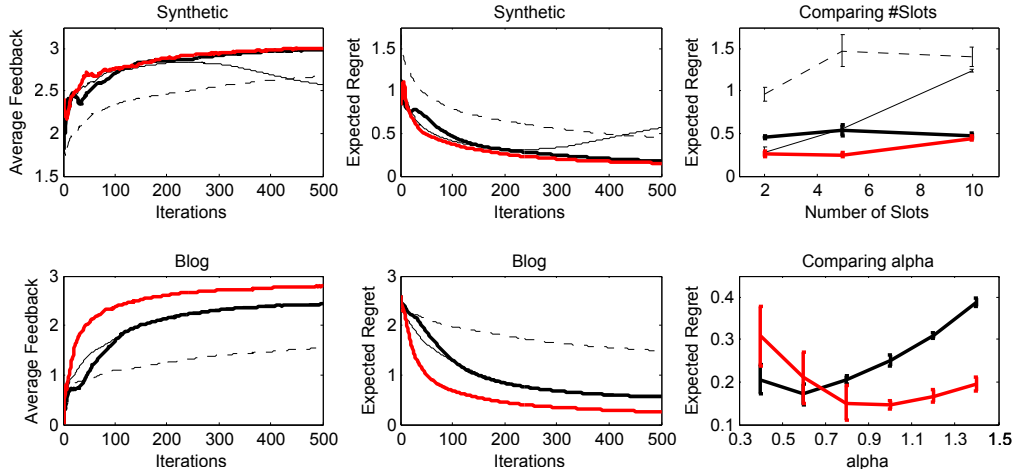

Figure 2: Simulation results comparing LSBGREEDY (red), RankLinUCB (black thick), Multiplicative Weighting (black thin), and $\epsilon$-Greedy (dashed thin). The middle column computes regret versus the clairvoyant greedy solution, and not $(1 - 1/e)OPT$. Unless specified, results are for $L = 5$.

were randomly generated using $d = 25$ topics, and $w^*$ was randomly generated and re-scaled so the most likely articles were liked with probability $\approx 75\%$. For the blog dataset, articles are represented using $d = 100$ topics generated using Latent Dirichlet Allocation [4], and $w^*$ was derived from a preliminary version of our user study. Our simulated user behaves according to the user model described in Section 3. We use probabilistic coverage (2) as the submodular basis functions.

**Competing Methods**. We compared LSBGREEDY against the following online learning algorithms. Note that all learning algorithms use the same underlying submodular utility model.

- Multiplicative Weighting (MW) as proposed in [9], which does not employ exploration.
- RankLinUCB, which combines the LinUCB algorithm [8, 20, 15, 7, 1] with Ranked Bandits [18, 22]. RankLinUCB is similar to LSBGREEDY except that it maintains a separate weight vector per slot since it employs a reduction to $L$ separate linear bandits (one per slot). In a sense, this is the natural application of existing approaches to our setting.[4]
- $\epsilon$-Greedy, which randomly explores with probability $\epsilon$, and exploits otherwise [15].

**Results**. Figure 2 shows a representative sample of our simulation results.[5] We see that both $\epsilon$-Greedy and Multiplicative Weighting achieve significantly worse results than LSBGREEDY. We also observe the performance of Multiplcative Weigthing diverge in the synthetic dataset, which is due to the fact that it does not employ exploration. RankLinUCB is more competitive, and achieves matching performance in the synthetic dataset. We also see that RankLinUCB is more sensitive to the choice of $\alpha$. Interestingly, both LSBGREEDY and RankLinUCB approach the same performance when recommending $L = 10$ articles. This can be explained by the user's interests being saturated by 10 articles, and suggests that the bound in Theorem 1 could potentially be further refined. Additional details can be found in Appendix B in the supplementary material.

### 5.2 User Studies

**Design**. The design of our study is similar to the personalization study conducted in [9]. We presented each user with ten articles per day over ten days from January 18, 2009 to January 27, 2009. Each day, the articles are selected using an interleaving of two policies (described below). The articles are displayed as a title with its contents viewable via a preview pane. The user is instructed

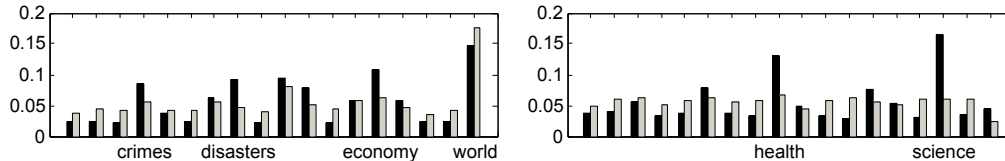

Figure 3: Displaying normalized learned preferences of LSBGREEDY (dark) and MW (light) for two user study sessions. In the left session, MW overfits to the "world" topic. In the right session, the user likes very few articles, and MW does not discover any topics that interest the user.

| COMPARISON | #SESSIONS | WIN/TIE/LOSE | GAIN PER DAY | % OF LIKES |
|---|---|---|---|---|
| LSBGREEDY vs Static Baseline | 24 | 24 / 0 / 0 | 1.07 | 63% (67%) |
| LSBGREEDY vs Mult. Weighting | 26 | 24 / 1 / 1 | 0.54 | 57% (63%) |
| LSBGREEDY vs RankLinUCB | 27 | 21 / 2 / 4 | 0.58 | 57% (61%) |

Table 1: User study comparing LSBGREEDY with competing algorithms. The parenthetical values in the last column are computed ignoring clicks on articles jointly recommended by both algorithms (see Section 5.2). All results are statistically significant with 95% confidence.

to briefly skim each article to get a sense of its content and, one by one, mark each article as "interested in reading in detail" (like), or "not interested" (dislike). As in [9], for each decision, the user is told to take into account the articles shown above in the current day, so as to capture the notion of incremental coverage. For example, a user might be interested in reading an article regarding the Middle East appearing at the top slot, and would mark it as "interested." However, if several very similar articles appear below it, the user may mark the subsequent articles as "not interested."

**Evaluation**. For each day, we generate an interleaving of recommendations from two algorithms. Interleaving allows us to make paired comparisons such that we simultaneously control for the particular user and particular day (certain days may contain more or less interesting content to the user than other days). Like other interleaving approaches [19], our approach maintains a notion of fairness so that both competing algorithms recommend the same amount of content. After each day, the user's feedback is collected and given to the two competing algorithms. Additional details of our experimental setup can be found in Appendix C in the supplementary material.

**Data**. In order to distinguish the gains of the algorithms from other effects (such as imperfections in the features, or having too high a dimension to converge), we performed dimensionality reduction. We created 18 genres (examples shown in Figure 3), labeled relevant articles and trained a model via linear regression for each genre. Note that many articles are relevant to multiple genres.

We compared LSBGREEDY against the static baseline (i.e., no personalization), Multiplicative Weighting (MW) from [9], and RankLinUCB. We evaluated each comparison setting using approximately twenty five participants, most of whom are graduate students or young professionals.

**Results**. Table 1 describes our results. We first aggregated per user, and then aggregated over all users. For each user, we computed three statistics: (1) whether LSBGREEDY won, tied, or lost in terms of total number of liked articles, (2) the difference in liked articles per day, and (3) the fraction of liked articles recommended by LSBGREEDY. Jointly recommended articles can be either counted as half to each algorithm or ignored (these results are shown in parentheticals in Table 1).

Overall, about 90% of users preferred recommendations by LSBGREEDY over the competing algorithms. On average, LSBGREEDY obtains about one additional liked article per day and 63% of all liked articles versus the static baseline, and about half an additional liked article per day and 57% of all liked articles versus the two competing learning algorithms. The gains we observe are all statistically significant with 95% confidence, and show that LSBGREEDY can be effective even when the assumptions in our theoretical analysis may not be satisfied.

Figure 3 shows the learned preferences by LSBGREEDY and MW on two sessions. Since MW does not employ exploration, it can either overfit to its previous experience and not find new topics that interest the user (left plot), or fail to discover any good topics (right plot). We do not include a comparison with RankLinUCB since it learns $L$ preference vectors, which are difficult to visualize.

# 6 Related Work

**Diversified Retrieval**. We are chiefly interested in training flexible submodular utility models, since such models yield practical algorithmic approaches. At one extreme are feature-free models that do not require training. However, such models are limited to unpersonalized settings that ignore context, such as recommending a global set of blogs to monitor [14]. On the other hand, methods that use feature-rich models typically either employ unsupervised training [24] or require fine-grained subtopic labels [25]. Such learning approaches cannot easily adapt to new domains. One exception is [9], whose proposed online learning approach does not incorporate exploration. As shown in our experiments, this significantly inhibits the learning ability of their approach.

Beyond submodular models of information coverage, other approaches include methods that balance relevance and novelty [5, 26, 6] and graph-based methods [27]. For such models, it remains a challenge to design provably efficient online learning algorithms.

**Bandit Learning**. From the perspective of our work, existing bandit approaches can be categorized along two dimensions: single-prediction versus set-prediction, and feature-based versus feature-free.

Most feature-based settings are designed to predict single results, rather than sets of results. Of such settings, the most relevant to ours is the linear stochastic bandits setting [8, 20, 15, 7, 1], which we build upon in our approach. One limitation here is the assumption of realizability – that the "true" user model lies within our class. It may be possible to develop more robust algorithms for our submodular bandits setting by building upon algorithms with more general guarantees (e.g., [2]).

Most set-based settings, such as bandit submodular optimization or the general bandit slate problem, assume a feature-free model [18, 22, 23, 12]. As such, performance is quantified relative to a fixed set of articles, which is not appropriate for many retrieval settings (e.g., news recommendation). One exception is [21], which assumes that document and user models lie within a metric space. However, it is unclear how to incorporate our submodular features into their setting.

# 7 Discussion of Limitations and Future Work

**Submodular Basis Features**. Our approach requires access to submodular basis functions as features. In practice these basis features are often derived using various topic modeling or dimensionality reduction techniques. However, the resulting features are almost always noisy or biased. Furthermore, one expects that different users will be better modeled using different basis features. As such, one important direction for future work is to learn the appropriate basis features from user feedback, which is similar to the setting of interactive topic modeling [11].

Moreover, user behavior is likely to be influenced by many factors beyond those well-modeled by submodular basis features. For example, the probability of the user liking a certain article could be influenced by the time of day, or day of the week. A more unified approach would be to incorporate both these standard features as well as submodular basis features in a joint model.

**Curse of Dimensionality**. The convergence rate of LSBGREEDY depends linearly on the number of features $d$ (which appears unavoidable without further assumptions). Thus, our approach may not be practical for settings that use a very large number of features. One possible extension is to jointly learn from multiple users simultaneously. If users tend to have similar preferences, then learning jointly from multiple users may yield convergence rates that are sub-linear in $d$.

# 8 Conclusion

We proposed an online learning setting for optimizing a general class of submodular functions. This setting is well-suited for modeling diversified retrieval systems that interactively learn from user feedback. We presented an algorithm, LSBGREEDY, and proved that it efficiently converges to a near-optimal model. We conducted simulations as well as user studies in the setting of news recommendation, and found that LSBGREEDY outperforms competing online learning approaches.

**Acknowledgements.** This work was funded in part by ONR (PECASE) N000141010672 and ONR Young Investigator Program N00014-08-1-0752. The authors also thank Khalid El-Arini, Joey Gonzalez, Sue Ann Hong, Jing Xiang, and the anonymous reviewers for their helpful comments.

## Footnotes

[1]In general, these features can represent any "nugget of information", such as a single word [24, 25, 9].

[2]E.g., the topics and coverage probabilities can be derived from a topic model such as LDA [4].

[3]Note that $w_t$ may have negative components, which would make $F(\cdot|w_t)$ not monotone submodular. However, regret is measured by $F(\cdot|w^*)$, which is monotone submodular. We show in our analysis that having negative components in $w_t$ does not hinder our ability to converge efficiently to $w^*$ in a regret sense.

[4]One can show that RankLinUCB achieves greedy regret (6) that grows as $\mathcal{O}(dL\sqrt{T})$ (ignoring log factors), which is a factor $\sqrt{L}$ worse than the regret guarantee of LSBGREEDY.

[5]For all methods, we find performance to be relatively stable w.r.t. the tuning parameters (e.g., $\alpha_t$ for LSBGREEDY). Unless specified, we set all parameters to values that achieve good results for their respective algorithms. In particular we set $\alpha_t = 1$ for LSBGREEDY, $\alpha_t = 0.6$ for RankLinUCB, $\beta = 0.9$ for MW, and $\epsilon = 0.1$ for $\epsilon$-Greedy. LSBGREEDY, RankLinUCB, and $\epsilon$-Greedy train linear models with regularization parameter $\lambda$, which we kept constant at $\lambda = 1$.

# References

[1] Y. Abbasi-Yadkori, D. Pal, and C. Szepesvari. Online least squares estimation with self-normalized processes: An application to bandit problems, 2011. http://arxiv.org/abs/1102.2670.

[2] J. Abernathy, E. Hazan, and A. Rakhlin. Competing in the dark: An efficient algorithm for bandit linear optimization. In *Conference on Learning Theory (COLT)*, 2008.

[3] R. Agrawal, S. Gollapudi, A. Halverson, and S. Ieong. Diversifying search results. In *ACM Conference on Web Search and Data Mining (WSDM)*, 2009.

[4] D. Blei, A. Ng, and M. Jordan. Latent dirichlet allocation. *Journal of Machine Learning Research (JMLR)*, 3:993–1022, 2003.

[5] J. Carbonell and J. Goldstein. The use of MMR, diversity-based re-ranking for reordering documents and producing summaries. In *ACM Conference on Information Retrieval (SIGIR)*, 1998.

[6] H. Chen and D. Karger. Less is more. In *ACM Conference on Information Retrieval (SIGIR)*, 2006.

[7] W. Chu, L. Li, L. Reyzin, and R. Schapire. Contextual bandits with linear payoff functions. In *Conference on Artificial Intelligence and Statistics (AISTATS)*, 2011.

[8] V. Dani, T. Hayes, and S. Kakade. Stochastic linear optimization under bandit feedback. In *Conference on Learning Theory (COLT)*, 2008.

[9] K. El-Arini, G. Veda, D. Shahaf, and C. Guestrin. Turning down the noise in the blogosphere. In *ACM Conference on Knowledge Discovery and Data Mining (KDD)*, 2009.

[10] U. Feige. A threshold of $\ln n$ for approximating set cover. *Journal of the ACM (JACM)*, 45(4):634–652, 1998.

[11] Y. Hu, J. Boyd-Graber, and B. Satinoff. Interactive topic modeling. In *Annual Meeting of the Association for Computational Linguistics (ACL)*, 2011.

[12] S. Kale, L. Reyzin, and R. Schapire. Non-stochastic bandit slate problems. In *Neural Information Processing Systems (NIPS)*, 2010.

[13] J. Langford and T. Zhang. The epoch-greedy algorithm for contextual multi-armed bandits. In *Neural Information Processing Systems (NIPS)*, 2007.

[14] J. Leskovec, A. Krause, C. Guestrin, C. Faloutsos, J. VanBriesen, and N. Glance. Cost-effective outbreak detection in networks. In *ACM Conference on Knowledge Discovery and Data Mining (KDD)*, 2007.

[15] L. Li, W. Chu, J. Langford, and R. Schapire. A contextual-bandit approach to personalized news article recommendation. In *World Wide Web Conference (WWW)*, 2010.

[16] L. Li, D. Wang, T. Li, D. Knox, and B. Padmanabhan. Scene: A scalable two-stage personalized news recommendation system. In *ACM Conference on Information Retrieval (SIGIR)*, 2011.

[17] G. Nemhauser, L. Wolsey, and M. Fisher. An analysis of the approximations for maximizing submodular set functions. *Mathematical Programming*, 14:265–294, 1978.

[18] F. Radlinski, R. Kleinberg, and T. Joachims. Learning diverse rankings with multi-armed bandits. In *International Conference on Machine Learning (ICML)*, 2008.

[19] F. Radlinski, M. Kurup, and T. Joachims. How does clickthrough data reflect retrieval quality? In *ACM Conference on Information and Knowledge Management (CIKM)*, 2008.

[20] P. Rusmevichientong and J. Tsitsiklis. Linearly parameterized bandits. *Mathematics of Operations Research*, 35(2):395–411, 2010.

[21] A. Slivkins, F. Radlinski, and S. Gollapudi. Learning optimally diverse rankings over large document collections. In *International Conference on Machine Learning (ICML)*, 2010.

[22] M. Streeter and D. Golovin. An online algorithm for maximizing submodular functions. In *Neural Information Processing Systems (NIPS)*, 2008.

[23] M. Streeter, D. Golovin, and A. Krause. Online learning of assignments. In *Neural Information Processing Systems (NIPS)*, 2009.

[24] A. Swaminathan, C. Mathew, and D. Kirovski. Essential pages. In *The IEEE/WIC/ACM International Conference on Web Intelligence (WI)*, 2009.

[25] Y. Yue and T. Joachims. Predicting diverse subsets using structural svms. In *International Conference on Machine Learning (ICML)*, 2008.

[26] C. Zhai, W. W. Cohen, and J. Lafferty. Beyond independent relevance: methods and evaluation metrics for subtopic retrieval. In *ACM Conference on Information Retrieval (SIGIR)*, 2003.

[27] X. Zhu, A. Goldberg, J. V. Gael, and D. Andrzejewski. Improving diversity in ranking using absorbing random walks. In *NAACL Conference on Human Language Technologies (HLT)*, 2007.

